# An Approximate Inference Approach for the PCA Reconstruction Error

**Manfred Opper**
Electronics and Computer Science
University of Southampton
Southampton, SO17 1BJ
`mo@ecs.soton.ac.uk`

## Abstract

The problem of computing a resample estimate for the reconstruction error in PCA is reformulated as an inference problem with the help of the replica method. Using the expectation consistent (EC) approximation, the intractable inference problem can be solved efficiently using only two variational parameters. A perturbative correction to the result is computed and an alternative simplified derivation is also presented.

## 1 Introduction

This paper was motivated by recent joint work with Ole Winther on approximate inference techniques (the expectation consistent (EC) approximation [1] related to Tom Minka's EP [2] approach) which allows us to tackle high–dimensional sums and integrals required for Bayesian probabilistic inference.

I was looking for a nice model on which I could test this approximation. It had to be simple enough so that I would not be bogged down by large numerical simulations. But it had to be nontrivial enough to be of at least modest interest to Machine Learning. With the somewhat unorthodox application of approximate inference to resampling in PCA I hope to be able to stress the following points:

- Approximate efficient inference techniques can be useful in areas of Machine Learning where one would not necessarily assume that they are applicable. This can happen when the underlying probabilistic model is not immediately visible but shows only up as a result a of mathematical transformation.

- Approximate inference methods can be highly robust allowing for analytic continuations of model parameters to the complex plane or even noninteger dimensions.

- It is not always necessary to use a large number of variational parameters in order to get reasonable accuracy.

- Inference methods could be systematically improved using perturbative corrections.

The work was also stimulated by previous joint work with Dörthe Malzahn [3] on resampling estimates for generalization errors of Gaussian process models and Supportvector–Machines.

## 2 Resampling estimators for PCA

*Principal Component Analysis* (PCA) is a well known and widely applied tool for data analysis. The goal is to project data vectors $\mathbf{y}$ from a typically high ($d$-) dimensional space into an optimally chosen lower ($q$-) dimensional linear space with $q << d$, thereby minimizing the expected projection error $\varepsilon = E||\mathbf{y} - P_q[\mathbf{y}]||^2$, where $P_q[\mathbf{y}]$ denotes the projection. $E$ stands for an expectation over the distribution of the data. In practice where the distribution is not available, one has to work with a data sample $D_0$ consisting of $N$ vectors $\mathbf{y}_k = (y_k(1), y_k(2), \ldots, y_k(d))^T$, $k = 1, \ldots, N$. We arrange these vectors into a ($d \times N$) data matrix $\mathbf{Y} = (\mathbf{y}_1, \mathbf{y}_2, \ldots, \mathbf{y}_N)$. Assuming centered data, the optimal subspace is spanned by the eigenvectors $\mathbf{u}_l$ of the $d \times d$ *data covariance matrix* $\mathbf{C} = \frac{1}{N}\mathbf{Y}\mathbf{Y}^T$ corresponding to the $q$ largest eigenvalues $\lambda_k$. We will assume that these correspond to all eigenvectors $\lambda_k > \lambda$ above some threshold value $\lambda$.

After computing the PCA projection, one would be interested in finding out if the computed subspace represents the data well by estimating the average projection error on *novel data* $\mathbf{y}$ (ie not contained in $D_0$) which are drawn from the same distribution.

Fixing the projection $P_q$, the error can be rewritten as

$$\mathcal{E} = \sum_{\lambda_l < \lambda} E \operatorname{Tr} \left[ \mathbf{y}\mathbf{y}^T \mathbf{u}_l \mathbf{u}_l^T \right] \tag{1}$$

where the expectation is only over $\mathbf{y}$ and the training data are fixed. The *training error* $\mathcal{E}_t = \sum_{\lambda_l < \lambda} \lambda_l^2$ can be obtained without knowledge of the distribution but will usually only give an optimistically biased estimate for $\mathcal{E}$.

### 2.1 A resampling estimate for the error

New artificial data samples $D$ of arbitrary size can be created by resampling a number of data points from $D_0$ with or without replacement. A simple choice would be to choose all data independently with the same probability $1/N$, but other possibilities can also be implemented within our formalism. Thus, some $\mathbf{y}_i$ in $D_0$ may appear multiple times in $D$ and others not at all. The idea of performing PCA on resampled data sets $D$ and testing on the remaining data $D_0 \backslash D$, motivates the following definition of a *resample averaged* reconstruction error

$$\mathcal{E}_r = \frac{1}{N_0} E_D \left[ \sum_{\mathbf{y}_i \notin D; \lambda_l < \lambda} \operatorname{Tr} \left( \mathbf{y}_i \mathbf{y}_i^T \mathbf{u}_l \mathbf{u}_l^T \right) \right] \tag{2}$$

as a proxy for $\mathcal{E}$. $E_D$ is the expectation over the resampling process. This is an estimator of the *bootstrap* type [3,4]. $N_0$ is the expected number of data in $D_0$ which are not contained in the random set $D$. The rest of the paper will discuss a method for efficiently approximating (2).

### 2.2 Basic formalism

We introduce "occupation numbers" $s_i$ which count how many times $\mathbf{y}_i$ is containd in $D$. We also introduce two matrices $\mathbf{D}$ and $\mathbf{C}$. $\mathbf{D}$ is a *diagonal random matrix*

$$\mathbf{D}_{ii} = D_i = \frac{1}{\mu\Gamma}(s_i + \epsilon\delta_{s_i,0}) \qquad \mathbf{C}(\epsilon) = \frac{\Gamma}{N}\mathbf{Y}\mathbf{D}\mathbf{Y}^T . \tag{3}$$

$\mathbf{C}(0)$ is proportional to the covariance matrix of the *resampled* data. $\mu$ is the sampling rate, i.e. $\mu N = E_D[\sum_i s_i]$ is the expexted number of data in $D$ (counting multiplicities). The

role of $\Gamma$ will be explained later. Using $\epsilon$, we can generate expressions that can be used in (2) to sum over the data which are not contained in the set $D$

$$\mathbf{C}'(0) = \frac{1}{\mu N} \sum_j \delta_{s_j,0} \mathbf{y}_j \mathbf{y}_j^T \ . \tag{4}$$

In the following $\lambda_k$ and $\mathbf{u}_k$ will always denote eigenvalues and eigenvectors of the data dependent (i.e. random) covariance matrix $\mathbf{C}(0)$.

The desired averages can be constructed from the $d \times d$ *matrix Green's function*

$$\mathbf{G}(\Gamma) = (\mathbf{C}(0) + \Gamma \mathbf{I})^{-1} = \sum_k \frac{\mathbf{u}_k \mathbf{u}_k^T}{\lambda_k + \Gamma} \tag{5}$$

Using the well known representation of the *Dirac* $\delta$ distribution given by $\delta(x) = \lim_{\eta \to 0^+} \Im \frac{1}{\pi(x - i\eta)}$ where $i = \sqrt{-1}$ and $\Im$ denotes the imaginary part, we get

$$\lim_{\eta \to 0^+} \frac{1}{\pi} \Im \, \mathbf{G}(\Gamma - i\eta) = \sum_k \mathbf{u}_k \mathbf{u}_k^T \delta \left( \lambda_k + \Gamma \right) \ . \tag{6}$$

Hence, we have

$$\mathcal{E}_r = \mathcal{E}_r^0 + \int_{0^+}^\lambda d\lambda' \ \varepsilon_r(\lambda') \tag{7}$$

where

$$\varepsilon_r(\lambda) = \frac{1}{\pi} \lim_{\eta \to 0^+} \Im \, \frac{1}{N_0} E_D \left[ \sum_j \delta_{s_j,0} \operatorname{Tr} \left( \mathbf{y}_j \mathbf{y}_j^T \mathbf{G}(-\lambda - i\eta) \right) \right] \tag{8}$$

defines the *error density* from all eigenvalues $> 0$ and $\mathcal{E}_r^0$ is the contribution from the eigenspace with $\lambda_k = 0$. The latter can also be easily expressed from $\mathbf{G}$ as

$$\mathcal{E}_r^0 = \lim_{\Gamma \to 0} \frac{1}{N_0} E_D \left[ \sum_j \delta_{s_j,0} \operatorname{Tr} \left( \mathbf{y}_j \mathbf{y}_j^T \Gamma \mathbf{G}(\Gamma) \right) \right] \tag{9}$$

We can also compute the resample averaged density of eigenvalues using

$$\rho(\lambda) = \frac{1}{\pi \mu N} \lim_{\eta \to 0^+} \Im \, E_D \left[ \operatorname{Tr} \mathbf{G}(-\lambda - i\eta) \right] \tag{10}$$

## 3   A Gaussian probabilistic model

The matrix Green's function for $\Gamma > 0$ can be generated from a Gaussian partition function $Z$. This is a well known construction in statistical physics, and has also been used within the NIPS community to study the distribution of eigenvalues for an average case analysis of PCA [5]. Its use for computing the expected reconstruction error is to my knowledge new.

With the $(N \times N)$ *kernel matrix* $\mathbf{K} = \frac{1}{N} \mathbf{Y}^T \mathbf{Y}$ we define the Gaussian partition function

$$
\begin{aligned}
Z &= \int d\mathbf{x} \, \exp \left[ -\frac{1}{2} \mathbf{x}^T \left( \mathbf{K}^{-1} + \mathbf{D} \right) \mathbf{x} \right] && (11) \\
&= |\mathbf{K}|^{\frac{1}{2}} \Gamma^{d/2} (2\pi)^{(N-d)/2} \int d^d \mathbf{z} \, \exp \left[ -\frac{1}{2} \mathbf{z}^T \left( \mathbf{C}(\epsilon) + \Gamma \mathbf{I} \right) \mathbf{z} \right] \ . && (12)
\end{aligned}
$$

$\mathbf{x}$ is an $N$ dimensional integration variable. The equality can be easily shown by expressing the integrals as determinants. [1] The first representation (11) is useful for computing the resampling average and the second one connects directly to the definition of the matrix Green's function $\mathbf{G}$. Note, that by its dependence on the kernel matrix $\mathbf{K}$, a generalization to $d = \infty$ dimensional feature spaces and *kernel PCA* is straightforward. The partition function can then be understood as a certain Gaussian process expectation. We will not discuss this point further.

The *free energy* $F = -\ln Z$ enables us to generate the following quantities

$$-2\frac{\partial \ln Z}{\partial \epsilon}\bigg|_{\epsilon=0} = \frac{1}{\mu N}\sum_{j=1}^{N}\delta_{s_j,0}\operatorname{Tr}\mathbf{y}_j\mathbf{y}_j^T\mathbf{G}(\Gamma) \tag{13}$$

$$-2\frac{\partial \ln Z}{\partial \Gamma} = \frac{d}{\Gamma} + \operatorname{Tr}\mathbf{G}(\Gamma) \tag{14}$$

where we have used (4) for (13). (13) will be used for the computation of (8) and (14) applies to the density of eigenvalues. Note that the definition of the partition function $Z$ requires that $\Gamma > 0$, whereas the application to the reconstruction error (7) needs negative values $\Gamma = -\lambda < 0$. Hence, an analytic continuation of end results must be performed.

## 4  Resampling average and replicas

(13) and (14) show that we can compute the desired resampling averages from the expected free energy $-E_D[\ln Z]$. This can be expressed using the "replica trick" of statistical physics (see e.g. [6]) using

$$E_D[\ln Z] = \lim_{n\to 0}\frac{1}{n}\ln E_D[Z^n]\,, \tag{15}$$

where one attempts an approximate computation of $E_D[Z^n]$ for *integer* $n$ and uses a continuation to real numbers at the end. The $n$ times replicated and averaged partition function (11) can be written in the form

$$Z^{(n)} \doteq E_D[Z^n] = \int dx\ \psi_1(x)\ \psi_2(x) \tag{16}$$

where we set $x \doteq (\mathbf{x}_1, \ldots, \mathbf{x}_n)$ and

$$\psi_1(x) = E_D\left[\exp\left\{-\frac{1}{2}\sum_{a=1}^{n}\mathbf{x}_a^T\mathbf{D}\mathbf{x}_a\right\}\right] \qquad \psi_2(x) = \exp\left[-\frac{1}{2}\sum_{a=1}^{n}\mathbf{x}_a^T\mathbf{K}^{-1}\mathbf{x}_a\right] \tag{17}$$

The *unaveraged* partition function $Z$ (11) is Gaussian, but the *averaged* $Z^{(n)}$ is not and usually intractable.

## 5  Approximate inference

To approximate $Z^{(n)}$, we will use the EC approximation recently introduced by Opper & Winther [1]. For this method we need two auxiliary distributions

$$p_1(x) = \frac{1}{Z_1}\psi_1(x)e^{-\Lambda_1 x^T x} \qquad p_0(x) = \frac{1}{Z_0}e^{-\frac{1}{2}\Lambda_0 x^T x}\,, \tag{18}$$

where $\Lambda_1$ and $\Lambda_0$ are "variational" parameters to be optimized. $p_1$ tries to mimic the intractable $p(x) \propto \psi_1(x)\ \psi_2(x)$, replacing the multivariate Gaussian $\psi_2$ by a simpler, i.e.

tractable diagonal one. One may think of using a general diagonal matrix $\Lambda_1$, but we will restrict ourselves in the present case to the simplest case of a spherical Gaussian with a *single parameter* $\Lambda_1$.

The strategy is to split $Z^{(n)}$ into a product of $Z_1$ and a term that has to be further approximated:

$$
\begin{aligned}
Z^{(n)} &= Z_1 \int dx \, p_1(x) \, \psi_2(x) \, e^{\Lambda_1 x^T x} \qquad (19) \\
&\approx Z_1 \int dx \, p_0(x) \, \psi_2(x) \, e^{\Lambda_1 x^T x} \equiv Z_{EC}^{(n)}(\Lambda_1, \Lambda_0) \, .
\end{aligned}
$$

The approximation replaces the intractable average over $p_1$ by a tractable one over $p_0$. To optimize $\Lambda_1$ and $\Lambda_0$ we argue as follows: We try to make $p_0$ as close as possible to $p_1$ by matching the moments $\langle x^T x \rangle_1 = \langle x^T x \rangle_0$. The index denotes the distribution which is used for averaging. By this step, $\Lambda_0$ becomes a function of $\Lambda_1$. Second, since the true partition function $Z^{(n)}$ is *independent* of $\Lambda_1$, we expect that a good approximation to $Z^{(n)}$ should be stationary with respect to variations of $\Lambda_1$. Both conditions can be expressed by the requirement that $\ln Z_{EC}^{(n)}(\Lambda_1, \Lambda_0)$ must be stationary with respect to variations of $\Lambda_1$ and $\Lambda_0$.

Within this EC approximation we can carry out the replica limit $E_D[\ln Z] \approx \ln Z_{EC} = \lim_{n \to 0} \frac{1}{n} \ln Z_{EC}^{(n)}$ and get after some calculations

$$
\begin{aligned}
-\ln Z_{EC} &= -E_D \left[ \ln \int d\mathbf{x} \, e^{-\frac{1}{2} \mathbf{x}^T (\mathbf{D} + (\Lambda_0 - \Lambda)\mathbf{I}) \mathbf{x}} \right] - \qquad (20) \\
&\quad - \ln \int d\mathbf{x} \, e^{-\frac{1}{2} \mathbf{x}^T (\mathbf{K}^{-1} + \Lambda \mathbf{I}) \mathbf{x}} + \ln \int d\mathbf{x} \, e^{-\frac{1}{2} \Lambda_0 \mathbf{x}^T \mathbf{x}}
\end{aligned}
$$

where we have set $\Lambda = \Lambda_0 - \Lambda_1$. Since the first Gaussian integral factorises, we can now perform the resampling average in (20) relatively easy for the case when all $s_j$'s in (3) are independent. Assuming e.g. *Poisson* probabilities $p(s) = e^{-\mu} \frac{\mu^s}{s!}$ gives a good approximation for the case of resampling $\mu N$ points with replacement.

The variational equations which make (20) stationary are

$$
E_D \left( \frac{1}{\Lambda_0 - \Lambda + D_i} \right) = \frac{1}{\Lambda_0} \qquad \frac{1}{N} \sum_k \frac{\omega_i}{1 + \omega_k \Lambda} = \frac{1}{\Lambda_0} \qquad (21)
$$

where $\omega_k$ are the eigenvalues of the matrix $\mathbf{K}$. The variational equations have to be solved in the region $\Gamma = -\lambda < 0$ where the original partition function does not exist. The resulting parameters $\Lambda_0$ and $\Lambda$ will usually come out as *complex* numbers.

## 6   Experiments

By eliminating the parameter $\Lambda_0$ from (21) it is possible to reduce the numerical computations to solving a nonlinear equation for a single *complex* parameter $\Lambda$ which can be solved easily and fast by a Newton method. While the analytical results are based on *Poisson* statistics, the simulations of random resampling was performed by choosing a *fixed* number (equal to the expected number of the Poisson distribution) of data at random with replacement.

The first experiment was for a set of data generated at random from a spherical Gaussian. To show that resampling maybe useful, we give on on the left hand side of Figure 1 the reconstruction error as a function of the value of $\lambda$ below which eigenvalues are dicarded.

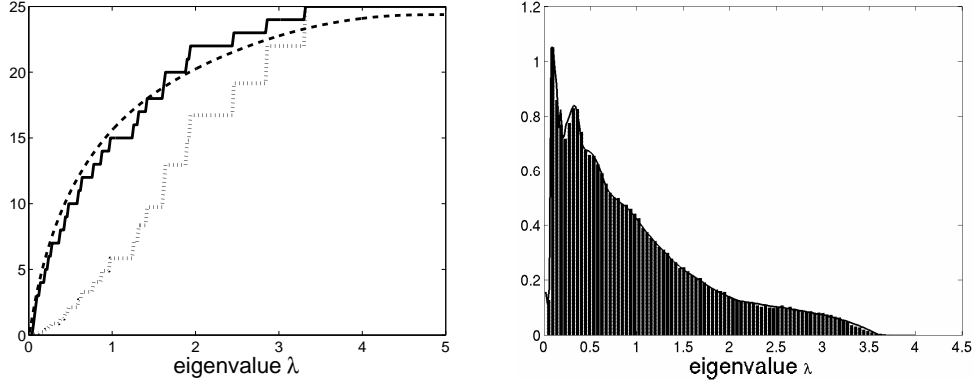

Figure 1: *Left:* Errors for PCA on $N = 32$ spherically Gaussian data with $d = 25$ and $\mu = 3$. Smooth curve: approximate resampled error estimate, upper step function: true error. Lower step function: Training error. *Right:* Comparison of EC approximation (line) and simulation (histogramme) of the resampled density of eigenvalues for $N = 50$ spherically Gaussian data of dimensionality $d = 25$. The sampling rate was $\mu = 3$.

The smooth function is the approximate resampling error ($3\times$ oversampled to leave not many data out of the samples) from our method. The upper step function gives the true reconstruction error (easy to calculate for spherical data) from (1). The lower step function is the training error. The right panel demonstrates the *accuracy* of the approximation on a similar set of data. We compare the analytically approximated density of states with the results of a true resampling experiment, where eigenvalues for many samples are counted into small bins. The theoretical curve follows closely the experiment.

Since the good accuracy might be attributed to the high symmetry of the toy data, we have also performed experiments on a set of $N = 100$ handwritten digits with $d = 784$. The results in Figure 2 are promising. Although the density of eigenvalues is more accurate than the resampling error, the latter comes still out reasonable.

## 7   Corrections

I will show next that the EC approximation can be augmented by a perturbation expansion. Going back to (19), we can write

$$\frac{Z^{(n)}}{Z_1} = \int dx \; p_1(x) \; \psi_2(x) \; e^{\Lambda_1 x^T x} = \int dx \; \psi_2(x) \; e^{\frac{1}{2}\Lambda x^T x} \left\{ \int \frac{dk}{(2\pi)^{Nn}} e^{-ik^T x} \chi(k) \right\}$$

where $\chi(k) \doteq \int dx \; p_1(x) e^{ik^T x}$ is the *characteristic function* of the density $p_1$ (18). $\ln \chi(k)$ is the *cumulant generating function*. Using the symmetries of the density $p_1$, we can perform a power series expansion of $\ln \chi(k)$, which starts with a quadratic term (second cumulant)

$$\ln \chi(k) = -\frac{M_2}{2} k^T k + R(k) \; , \tag{22}$$

where $M_2 = \langle \mathbf{x}_a^T \mathbf{x}_a \rangle_1$. It can be shown that if we neglect $R(k)$ (containing the higher order cumulants) and carry out the integral over $k$, we end up replacing $p_1$ by a simpler Gaussian $p_0$ with matching moments $M_2$, i.e. the EC approximation. Higher order corrections to the free energy $-E_D[\ln Z] = -\ln Z_{EC} + \Delta F_1 + \ldots$ can be obtained perturbatively by writing $\chi(k) = e^{-\frac{M_2}{2} k^T k}(1 + R(k) + \ldots)$. This expansion is similar in spirit to *Edgeworth*

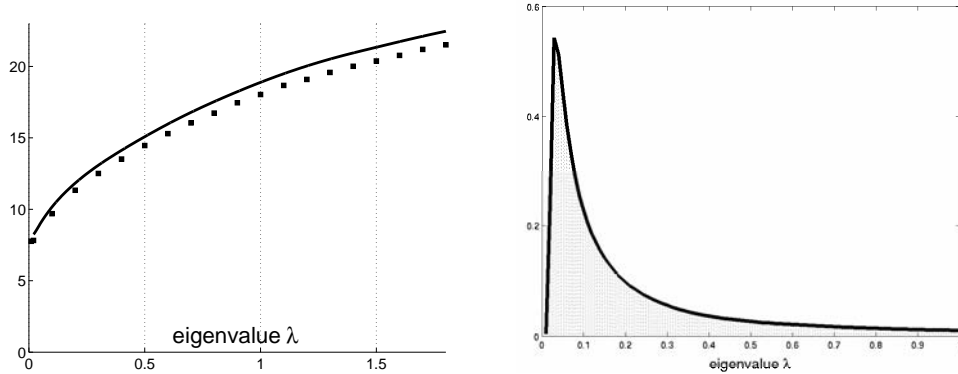

Figure 2: *Left:* Resampling error ($\mu = 1$) for PCA on a set of 100 handwritten digits ("5") with $d = 784$. The approximation (line) for $\mu = 1$ is compared with simulations of the random resampling. *Right:* Resampled density of eigenvalues for the same data set. Only the nonzero eigenvalues are shown.

*expansions* in statistics. The present case is more complicated by the extra dimensions introduced by the *replicating* of variables and the limit $n \to 0$. After a lengthy calculation one finds for the lowest order correction (containing the monomials in $k$ of order $4$) to the free energy:

$$\Delta F_1 = -\frac{1}{4} E_D \left( \frac{\Lambda_0}{\Lambda_0 - \Lambda + D_i} - 1 \right)^2 \times \sum_i \left( \Lambda_0 \left( \mathbf{K}^{-1} + \Lambda \mathbf{I} \right)_{ii}^{-1} - 1 \right)^2 \qquad (23)$$

I illustrate the effect of $\Delta F_1$ on a correction to the reconstruction error in the "zero–subspace" using (9) and (13) for the digit data as a function of $\mu$. Resampling used the Poisson approximation. The left panel of Figure 3 demonstrates that the true correction is fairly small. The right panel shows that the lowest order term $\Delta F_1$ accounts for a major part of the true correction when $\mu < 3$. The strong underestimation for larger $\mu$ needs further investigation.

## 8 The calculation without replicas

Knowing with hindsight how the final EC result (20) looks like, we can rederive it using another method which does not rely on the "replica trick". We first write down an exact expression for $-\ln Z$ *before* averaging. Expressing Gaussian integrals by determinants yields

$$-\ln Z = -\ln \int d\mathbf{x} \, e^{-\frac{1}{2} \mathbf{x}^T (\mathbf{D} + (\Lambda_0 - \Lambda)\mathbf{I}) \mathbf{x}} - \ln \int d\mathbf{x} \, e^{-\frac{1}{2} \mathbf{x}^T (\mathbf{K}^{-1} + \Lambda \mathbf{I}) \mathbf{x}} + \quad (24)$$

$$+ \ln \int d\mathbf{x} \, e^{-\frac{1}{2} \Lambda_0 \mathbf{x}^T \mathbf{x}} + \frac{1}{2} \ln \det(\mathbf{I} + \mathbf{r})$$

where the matrix $\mathbf{r}$ has elements $\mathbf{r}_{ij} = \left( 1 - \frac{\Lambda_0}{\Lambda_0 - \Lambda + D_i} \right) \left( \Lambda_0 \left( \mathbf{K}^{-1} + \Lambda \mathbf{I} \right)^{-1} - \mathbf{I} \right)_{ij}$. The EC approximation is obtained by simply neglecting $\mathbf{r}$. Corrections to this are found by expanding

$$\ln \det \left( \mathbf{I} + \mathbf{r} \right) = \operatorname{Tr} \ln \left( \mathbf{I} + \mathbf{r} \right) = \sum_{k=1}^{\infty} \frac{(-1)^{k+1}}{k} \operatorname{Tr} \left( \mathbf{r}^k \right) \qquad (25)$$

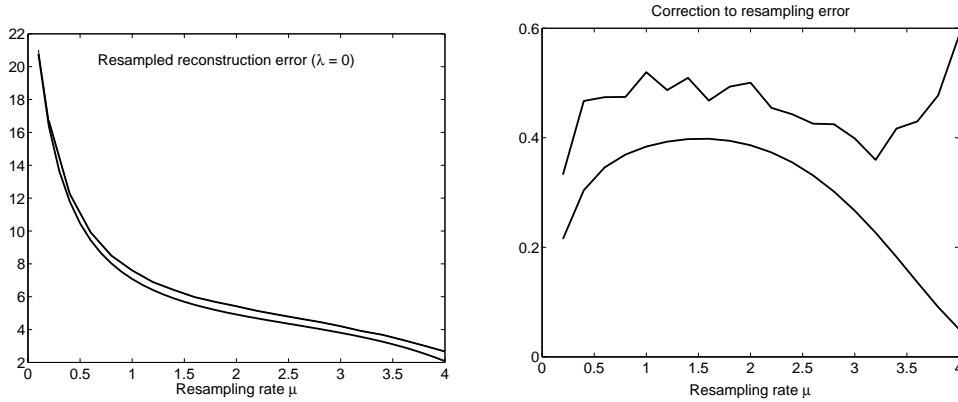

Figure 3: *Left:* Resampling error $\mathcal{E}_r^0$ from the $\lambda = 0$ subspace as a function of resampling rate for the digits data. The approximation (lower line) is compared with simulations of the random resampling (upper line). *Right:* The difference between approximation and simulations (upper curve) and its estimate (lower curve) from the perturbative correction (23).

The first order term in the expansion (25) vanishes after averaging (see (21)) and the second order term gives exactly the correction of the cumulant method (23).

## 9 Outlook

It will be interesting to extend the perturbative framework for the computation of corrections to inference approximations to other, more complex models. However, our results indicate that the use and convergence of such perturbation expansion needs to be critically investigated and that the lowest order may not always give a clear indication of the accuracy of the approximation. The alternative derivation for our simple model could present an interesting ground for testing these ideas.

### Acknowledgments

I would like to thank Ole Winther for the great collaboration on the EC approximation.

## Footnotes

[1] If $\mathbf{K}$ has zero eigenvalues, a division of $Z$ by $|\mathbf{K}|^{\frac{1}{2}}$ is necessary. This additive renormalization of the free energy $-\ln Z$ will not influence the subsequent computations.

### References

[1] Manfred Opper and Ole Winther. Expectation consistent free energies for approximate inference. In *NIPS 17*, 2005.

[2] T. P. Minka. Expectation propagation for approximate Bayesian inference. In *UAI 2001*, pages 362–369, 2001.

[3] D. Malzahn and M. Opper. An approximate analytical approach to resampling averages. *Journal of Machine Learning Research*, pages 1151–1173, 2003.

[4] B. Efron, R. J. Tibshirani. *An Introduction to the Bootstrap.* Monographs on Statistics and Applied Probability 57, Chapman & Hall, 1993.

[5] D. C. Hoyle and M. Rattray  Limiting form of the sample covariance matrix eigenspectrum in PCA and kernel PCA. In *NIPS 16*, 2003.

[6] A. Engel and C. Van den Broeck, *Statistical Mechanics of Learning* (Cambridge University Press, 2001).
